# No-regret Algorithms for Online Convex Programs

**Geoffrey J. Gordon**
Department of Machine Learning
Carnegie Mellon University
Pittsburgh, PA 15213
ggordon@cs.cmu.edu

## Abstract

Online convex programming has recently emerged as a powerful primitive for designing machine learning algorithms. For example, OCP can be used for learning a linear classifier, dynamically rebalancing a binary search tree, finding the shortest path in a graph with unknown edge lengths, solving a structured classification problem, or finding a good strategy in an extensive-form game. Several researchers have designed no-regret algorithms for OCP. But, compared to algorithms for special cases of OCP such as learning from expert advice, these algorithms are not very numerous or flexible. In learning from expert advice, one tool which has proved particularly valuable is the correspondence between no-regret algorithms and convex potential functions: by reasoning about these potential functions, researchers have designed algorithms with a wide variety of useful guarantees such as good performance when the target hypothesis is sparse. Until now, there has been no such recipe for the more general OCP problem, and therefore no ability to tune OCP algorithms to take advantage of properties of the problem or data. In this paper we derive a new class of no-regret learning algorithms for OCP. These *Lagrangian Hedging* algorithms are based on a general class of potential functions, and are a direct generalization of known learning rules like weighted majority and external-regret matching. In addition to proving regret bounds, we demonstrate our algorithms learning to play one-card poker.

## 1 Introduction

In a sequence of trials we must pick hypotheses $y_1, y_2, \ldots \in \mathcal{Y}$. After we choose $y_t$, the correct answer is revealed as a convex loss function $\ell_t(y_t)$.[1] Just before seeing the $t^{\text{th}}$ example, our total loss is therefore $L_t = \sum_{i=1}^{t-1} \ell_i(y_i)$. If we had predicted using some fixed hypothesis $y$ instead, then our loss would have been $\sum_{i=1}^{t-1} \ell_i(y)$. Our total *regret* at time $t$ is the difference between these two losses, with positive regret meaning that we would have preferred $y$ to our actual plays:

$$\rho_t(y) = L_t - \sum_{i=1}^{t-1} \ell_i(y) \qquad \rho_t = \sup_{y \in \mathcal{Y}} \rho_t(y)$$

We assume that $\mathcal{Y}$ is a compact convex subset of $\mathbb{R}^d$ that has at least two elements. In classical no-regret algorithms such as weighted majority, $\mathcal{Y}$ is a simplex: the corners of $\mathcal{Y}$ represent pure actions, the interior points of $\mathcal{Y}$ represent probability distributions over pure actions, and the number of corners $n$ is the same as the number of dimensions $d$. In a more general OCP, $\mathcal{Y}$ may have

many more extreme points than dimensions, $n \gg d$. For example, $\mathcal{Y}$ could be a convex set like $\{y \mid Ay = b, y \geq 0\}$ for some matrix $A$ and vector $b$, or it could even be a sphere.

The shape of $\mathcal{Y}$ captures the structure in our prediction problem. Each point in $\mathcal{Y}$ is a separate hypothesis, but the losses of different hypotheses are related to each other because they are all embedded in the common representation space $\mathbb{R}^d$. While we could run a standard no-regret algorithm such as weighted majority on a structured $\mathcal{Y}$ by giving it hypotheses corresponding to the extreme points $c_1 \ldots c_n$ of $\mathcal{Y}$, this transformation would lose the connections among hypotheses (with a corresponding loss in runtime and generalization ability).

Our algorithms below are stated in terms of linear loss functions, $\ell_t(y) = c_t \cdot y$. If $\ell_t$ is nonlinear but convex, we can substitute the derivative at the current prediction, $\partial \ell_t(y_t)$, for $c_t$, and our regret bounds will still hold (see [1, p. 53]). We will write $\mathcal{C}$ for the set of possible gradient vectors $c_t$.

## 2 Related Work

A large number of researchers have studied online prediction in general and OCP in particular. The OCP problem dates back to Hannan in 1957 [2]. The name "online convex programming" is due to Zinkevich [3], who gave a clever gradient-descent algorithm. A similar algorithm and a weaker bound were presented somewhat earlier in [1]: that paper's GGD algorithm, using potential function $\ell_0(w) = k\|w\|_2^2$, is equivalent to Zinkevich's "lazy projection" with a fixed learning rate. Another clever algorithm for OCP was presented by Kalai and Vempala [4].

Compared to the above papers, the most important contribution of the current paper is its generality: no previous family of OCP algorithms can use as flexible a class of potential functions. As an illustration of the importance of this generality, consider the problem of learning from expert advice. Well-known regret bounds for this problem are logarithmic in the number of experts (e.g., [5]); no previous bounds for general OCP algorithms are sublinear in the number of experts, but logarithmic bounds follow directly as a special case of our results [6, sec. 8.1.2]. Despite this generality, our core result, Thm. 4 below, takes only half a dozen short equations to prove.

From the online prediction literature, the closest related work is that of Cesa-Bianchi and Lugosi [7], which follows in the tradition of an algorithm and proof by Blackwell [8]. Cesa-Bianchi and Lugosi consider choosing predictions from an essentially-arbitrary decision space and receiving outcomes from an essentially-arbitrary outcome space. Together a decision and an outcome determine how a marker $R^t \in \mathbb{R}^d$ will move. Given a potential function $G$, they present algorithms which keep $G(R_t)$ from growing too quickly. This result is similar in flavor to our Thm. 4, and both Thm. 4 and the results of Cesa-Bianchi and Lugosi are based on Blackwell-like conditions. In fact, our Thm. 4 can be thought of as the first generalization of well-known online learning results such as Cesa-Bianchi and Lugosi's to online convex programming.

The main differences between the Cesa-Bianchi–Lugosi results and ours are the restrictions on their potential functions. They write their potential function as $G(u) = f(\Phi(u))$; they require $\Phi$ to be additive (that is, $\Phi(u) = \sum_i \phi_i(u_i)$ for one-dimensional functions $\phi_i$), nonnegative, and twice differentiable, and they require $f : \mathbb{R}^+ \mapsto \mathbb{R}^+$ to be increasing, concave, and twice differentiable. These restrictions rule out many of the potential functions used here, and in fact they rule out most online convex programming problems. The most restrictive requirement is additivity; for example, when defining potentials for OCPs via Eq. (7) below, unless the set $\bar{\mathcal{Y}}$ can be factored as $\bar{\mathcal{Y}}_1 \times \bar{\mathcal{Y}}_2 \times \ldots \times \bar{\mathcal{Y}}_N$ the potentials are generally not expressible as $f(\Phi(u))$ for additive $\Phi$.

During the preparation of this manuscript, we became aware of the recent work of Shalev-Shwartz and Singer [9]. This work generalizes some of the theorems in [6] and provides a very simple and elegant proof technique for algorithms based on convex potential functions. However, it does not consider the problem of defining appropriate potential functions for the feasible regions of OCPs (as discussed in Sec. 5 below and in more detail in [6]); finding such functions is an important requirement for applying potential-based algorithms to OCPs.

In addition to the general papers above, there are many no-regret algorithms for specific OCPs, such as predicting as well as the best pruning of a decision tree [10], reorganizing a binary search tree so that frequent items are near the root [4], and picking paths in a graph with unknown edge costs [11].

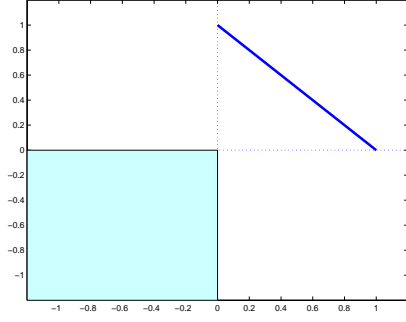

```
s₁ ← 0
for t ← 1, 2, . . .
    ȳ_t ← f(s_t)                           (*)
    if ȳ_t · u > 0 then
        y_t ← ȳ_t/(ȳ_t · u)
    else
        y_t ← arbitrary element of 𝒴
    fi
    Observe c_t, compute s_{t+1} from (1)
end
```

Figure 1: A set $\mathcal{Y} = \{y_1 + y_2 = 1, y \geq 0\}$ (thick dark line) and its safe set $\mathcal{S}$ (light shaded region).

Figure 2: The gradient form of the Lagrangian Hedging algorithm.

## 3 Regret Vectors

Lagrangian Hedging algorithms maintain their state in a *regret vector*, $s_t$, defined by the recursion

$$s_{t+1} = s_t + (y_t \cdot c_t)u - c_t \tag{1}$$

with the base case $s_1 = 0$. Here $u$ is an arbitrary vector which satisfies $y \cdot u = 1$ for all $y \in \mathcal{Y}$. (If necessary we can append a constant element to each $y$ so that such a $u$ exists.) The regret vector contains information about our actual losses and the gradients of our loss functions: from $s_t$ we can find our regret versus any $y$ as follows. (This property justifies the name "regret vector.")

$$y \cdot s_t = \sum_{i=1}^{t-1}(y_i \cdot c_i)y \cdot u - \sum_{i=1}^{t-1} y \cdot c_i = L_t - \sum_{i=1}^{t-1} y \cdot c_i = \rho_t(y)$$

We can define a *safe set*, in which our regret is guaranteed to be nonpositive:

$$\mathcal{S} = \{s \mid (\forall y \in \mathcal{Y})\, y \cdot s \leq 0\} \tag{2}$$

The goal of the Lagrangian Hedging algorithm is to keep its regret vector $s_t$ near the safe set $\mathcal{S}$. $\mathcal{S}$ is a convex cone: it is closed under positive linear combinations of its elements. And, it is *polar* [12] to the cone of unnormalized hypotheses:

$$\mathcal{S}^\perp = \bar{\mathcal{Y}} \equiv \{\lambda y \mid y \in \mathcal{Y},\, \lambda \geq 0\} \tag{3}$$

## 4 The Main Algorithm

We will present the general LH algorithm first, then (in Sec. 5) a specialization which is often easier to implement. The two versions are called the *gradient form* and the *optimization form*. The gradient form is shown in Fig. 2. At each step it chooses its play based on the current regret vector $s_t$ (Eq. (1)) and a closed convex potential function $F(s) : \mathbb{R}^d \mapsto \mathbb{R}$ with subgradient $f(s) : \mathbb{R}^d \mapsto \mathbb{R}^d$. This potential function is what distinguishes one instance of the LH algorithm from another. $F(s)$ should be small when $s$ is in the safe set, and large when $s$ is far from the safe set.

For example, suppose that $\mathcal{Y}$ is the probability simplex in $\mathbb{R}^d$, so that $\mathcal{S}$ is the negative orthant in $\mathbb{R}^d$. (This choice of $\mathcal{Y}$ would be appropriate for playing a matrix game or predicting from expert advice.) For this $\mathcal{Y}$, two possible potential functions are

$$F_1(s) = \ln \sum_i e^{\eta s_i} - \ln d \qquad F_2(s) = \sum_i [s_i]_+^2/2$$

where $\eta > 0$ is a learning rate and $[s]_+ = \max(s, 0)$. The potential $F_1$ leads to the Hedge [5] and weighted majority [13] algorithms, while the potential $F_2$ results in external-regret matching [14, Theorem B]. For more examples of useful potential functions, see [6].

To ensure the LH algorithm chooses legal hypotheses $y_t \in \mathcal{Y}$, we require the following (note the constant 0 is arbitrary; any other $k$ would work as well)

$$F(s) \leq 0 \qquad \forall s \in \mathcal{S} \tag{4}$$

**Theorem 1** *The LH algorithm is well-defined: define $\mathcal{S}$ as in (2) and fix a finite convex potential function $F$. If $F(s) \leq 0$ for all $s \in \mathcal{S}$, then the LH algorithm picks hypotheses $y_t \in \mathcal{Y}$ for all $t$.*

(Omitted proofs are given in [6].) We can also define a version of the LH algorithm with an adjustable learning rate: replacing $F(s)$ with $F(\eta s)$ is equivalent to updating $s_t$ with learning rate $\eta$. Adjustable learning rates will help us obtain regret bounds for some classes of potentials.

## 5  The Optimization Form

Even if we have a convenient representation of our hypothesis space $\mathcal{Y}$, it may not be easy to work directly with the safe set $\mathcal{S}$. In particular, it may be difficult to define, evaluate, and differentiate a potential function $F$ which has the necessary properties. To avoid these difficulties, we can work with an alternate form of the LH algorithm. This form, called the *optimization form*, defines $F$ in terms of a simpler function $W$ which we will call the *hedging function*. It uses the same pseudocode as the gradient form (Fig. 2), but on each step it computes $F$ and $\partial F$ by solving an optimization problem involving $W$ and the hypothesis set $\mathcal{Y}$ (Eq. (8) below).

For example, two possible hedging functions are

$$W_1(\bar{y}) = \begin{cases} \sum_i \bar{y}_i \ln \bar{y}_i + \ln d & \text{if } \bar{y} \geq 0, \sum_i \bar{y}_i = 1 \\ \infty & \text{otherwise} \end{cases} \tag{5}$$

$$W_2(\bar{y}) = \sum_i \bar{y}_i^2 / 2 \tag{6}$$

If $\mathcal{Y}$ is the probability simplex in $\mathbb{R}^d$, it will turn out that $W_1(\bar{y}/\eta)$ and $W_2(\bar{y})$ correspond to the potentials $F_1$ and $F_2$ from Section 4 above. So, these hedging functions result in the weighted majority and external-regret matching algorithms. For an example where the hedging function is easy to write analytically but the potential function is much more complicated, see Sec. 8 or [6].

The optimization form of the LH algorithm using hedging function $W$ is defined to be equivalent to the gradient form using

$$F(s) = \sup_{\bar{y} \in \bar{\mathcal{Y}}} (s \cdot \bar{y} - W(\bar{y})) \tag{7}$$

Here $\bar{\mathcal{Y}}$ is defined as in (3).[2] To implement the LH algorithm using the $F$ of Eq. (7), we need an efficient way to compute $\partial F$. As Thm. 2 below shows, there is always a $\bar{y}$ which satisfies

$$\bar{y} \in \arg\max_{\bar{y} \in \bar{\mathcal{Y}}} (s \cdot \bar{y} - W(\bar{y})) \tag{8}$$

and any such $\bar{y}$ is an element of $\partial F$. So, the optimization form of the LH algorithm uses the same pseudocode as the gradient form (Fig. 2), but uses Eq. (8) with $s = s_t$ to compute $\bar{y}_t$ in line $(*)$.

To gain an intuition for Eqs. (7–8), consider the example of external-regret matching. Since $\mathcal{Y}$ is the unit simplex in $\mathbb{R}^d$, $\bar{\mathcal{Y}}$ is the positive orthant in $\mathbb{R}^d$. So, with $W_2(\bar{y}) = \|\bar{y}\|_2^2/2$, the optimization problem (8) will be equivalent to

$$\bar{y} = \arg\min_{\bar{y} \in \mathbb{R}_+^d} \frac{1}{2} \|s - \bar{y}\|_2^2$$

That is, $\bar{y}$ is the projection of $s$ onto $\mathbb{R}_+^d$ by minimum Euclidean distance. It is not hard to verify that this projection replaces the negative elements of $s$ with zeros, $\bar{y} = [s]_+$. Substituting this value for $\bar{y}$ back into (7) and using the fact that $s \cdot [s]_+ = [s]_+ \cdot [s]_+$, the resulting potential function is

$$F_2(s) = s \cdot [s]_+ - \sum_i [s_i]_+^2/2 = \sum_i [s_i]_+^2/2$$

as claimed above. This potential function is the standard one for analyzing external-regret matching.

**Theorem 2** *Let $W$ be convex, $\operatorname{dom} W \cap \bar{\mathcal{Y}}$ be nonempty, and $W(\bar{y}) \geq 0$ for all $\bar{y}$. Suppose the sets $\{\bar{y} \mid W(\bar{y}) + s \cdot \bar{y} \leq k\}$ are compact for all $s$ and $k$. Define $F$ as in (7). Then $F$ is finite and $F(s) \leq 0$ for all $s \in \mathcal{S}$. And, the optimization form of the LH algorithm using the hedging function $W$ is equivalent to the gradient form of the LH algorithm with potential function $F$.*

# 6    Theoretical Results

Our main theoretical results are regret bounds for the LH algorithm. The bounds depend on the curvature of our potential $F$, the size of the hypothesis set $\mathcal{Y}$, and the possible slopes $\mathcal{C}$ of our loss functions. Intuitively, $F$ must be neither too curved nor too flat on the scale of the updates to $s_t$ from Eq. (1): if $F$ is too curved then $\partial F$ will change too quickly and our hypothesis $y_t$ will jump around a lot, while if $F$ is too flat then we will not react quickly enough to changes in regret.

We will state our results for the gradient form of the LH algorithm. For the optimization form, essentially the same results hold, but the constants are defined in terms of the hedging function instead. Therefore, we never need to work with (or even be able to write down) the corresponding potential function. For more details, see [6]. One result which is slightly tricky to carry over is tuning learning rates. The choice of learning rate below and the resulting bound are the same as for the gradient form, but the implementation is slightly different: to set a learning rate $\eta > 0$, we replace $W(\bar{y})$ with $W(\bar{y}/\eta)$.

We will need upper and lower bounds on $F$. We will assume

$$F(s + \Delta) \leq F(s) + \Delta \cdot f(s) + C\|\Delta\|^2 \tag{9}$$

for all regret vectors $s$ and increments $\Delta$, and

$$[F(s) + A]_+ \geq \inf_{s' \in \mathcal{S}} B\|s - s'\|^p \tag{10}$$

for all $s$. Here $\|\cdot\|$ is an arbitrary finite norm, and $A \geq 0$, $B > 0$, $C > 0$, and $1 \leq p \leq 2$ are constants. Eq. (9), together with the convexity of $F$, implies that $F$ is differentiable and $f$ is its gradient; the LH algorithm is applicable if $F$ is not differentiable, but its regret bounds are weaker.

We will bound the size of $\mathcal{Y}$ by assuming that

$$\|y\|_\circ \leq M \tag{11}$$

for all $y$ in $\mathcal{Y}$. Here, $\|\cdot\|_\circ$ is the dual of the norm used in Eq. (9) [12].

The size of our update to $s_t$ (in Eq. (1)) depends on the hypothesis set $\mathcal{Y}$, the cost vector set $\mathcal{C}$, and the vector $u$. We have already bounded $\mathcal{Y}$; rather than bounding $\mathcal{C}$ and $u$ separately, we will assume that there is a constant $D$ so that

$$E(\|s_{t+1} - s_t\|^2 \mid s_t) \leq D \tag{12}$$

Here the expectation is taken with respect to our choice of hypothesis, so the inequality must hold for all possible values of $c_t$. (The expectation is only necessary if we randomize our choice of hypothesis, as would happen if $\mathcal{Y}$ is the convex hull of some non-convex set. If interior points of $\mathcal{Y}$ are valid plays, we need not randomize, so we can drop the expectation in (12) and below.)

Our theorem then bounds our regret in terms of the above constants. Since the bounds are sublinear in $t$, they show that Lagrangian Hedging is a no-regret algorithm when we choose an appropriate potential $F$.

**Theorem 3** *Suppose the potential function $F$ is convex and satisfies Eqs. (4), (9) and (10). Suppose that the problem definition is bounded according to (11) and (12). Then the LH algorithm (Fig. 2) achieves expected regret*

$$E(\rho_{t+1}(y)) \leq M((tCD + A)/B)^{1/p} = O(t^{1/p})$$

*versus any hypothesis $y \in \mathcal{Y}$.*

*If $p = 1$ the above bound is $O(t)$. But, suppose that we know ahead of time the number of trials $t$ we will see. Define $G(s) = F(\eta s)$, where*

$$\eta = \sqrt{A/(tCD)}$$

*Then the LH algorithm with potential $G$ achieves regret*

$$E(\rho_{t+1}(y)) \leq (2M/B)\sqrt{tACD} = O(\sqrt{t})$$

*for any hypothesis $y \in \mathcal{Y}$.*

The full proof of Thm. 3 appears in [6]; here, we sketch the proof of one of the most important intermediate results. Thm. 4 shows that, if we can guarantee $E(s_{t+1} - s_t) \cdot \partial F(t) \leq 0$, then $F(s_t)$ cannot grow too quickly. This result is analogous to Blackwell's approachability theorem: since the level sets of $F$ are related to $\mathcal{S}$, we will be able to show $s_t / t \to \mathcal{S}$, implying no regret.

**Theorem 4 (Gradient descent)** *Let $F(s)$ and $f(s)$ satisfy Equation (9) with seminorm $\|\cdot\|$ and constant $C$. Let $x_0, x_1, \ldots$ be a sequence of random vectors. Write $s_t = \sum_{i=0}^{t-1} x_i$, and let $D$ be a constant so that $E(\|x_t\|^2 \mid s_t) \leq D$. Suppose that, for all $t$, $E(x_t \cdot f(s_t) \mid s_t) \leq 0$. Then for all $t$,*

$$E(F(s_{t+1}) \mid s_1) - F(s_1) \leq tCD$$

PROOF: The proof is by induction: for $t \geq 2$, assume $E(F(s_t) \mid s_1) \leq F(s_1) + (t-1)CD$. (It is obvious that the base case holds for $t = 1$.) Then:

$$
\begin{aligned}
F(s_{t+1}) &= F(s_t + x_t) \\
&\leq F(s_t) + x_t \cdot f(s_t) + C\|x_t\|^2 \\
E(F(s_{t+1}) \mid s_t) &\leq F(s_t) + CD \\
E(F(s_{t+1}) \mid s_1) &\leq E(F(s_t) \mid s_1) + CD \\
E(F(s_{t+1}) \mid s_1) &\leq F(s_1) + (t-1)CD + CD
\end{aligned}
$$

which is the desired result. □

## 7  Examples

The classical applications of no-regret algorithms are learning from expert advice and learning to play a repeated matrix game. These two tasks are essentially equivalent, since they both use the probability simplex $\mathcal{Y} = \{y \mid y \geq 0, \sum_i y_i = 1\}$ for their hypothesis set. This choice of $\mathcal{Y}$ simplifies the required algorithms greatly; with appropriate choices of potential functions, it can be shown that standard no-regret algorithms such as Freund and Schapire's Hedge [5], Littlestone and Warmuth's weighted majority [13], and Hart and Mas-Colell's external-regret matching [14, Theorem B] are all special cases of the LH algorithm.

A large variety of other online prediction problems can also be cast in our framework. These problems include path planning when costs are chosen by an adversary [11], planning in a Markov decision process when costs are chosen by an adversary [15], online pruning of a decision tree [16], and online balancing of a binary search tree [4]. More uses of online convex programming are given in [1, 3, 4]. In each case the bounds for the LH algorithm will be polynomial or better in the dimensionality of the appropriate hypothesis set and sublinear in the number of trials.

## 8  Experiments

To demonstrate that our theoretical bounds translate to good practical performance, we implemented the LH algorithm with the potential function $W_2$ from (6) and used it to learn policies for the game of one-card poker. (The hypothesis space for this learning problem is the set of *sequence weight vectors*, which is convex because one-card poker is an extensive-form game [17].)

In one-card poker, two players (called the *gambler* and the *dealer*) each ante \$1 and receive one card from a 13-card deck. The gambler bets first, adding either \$0 or \$1 to the pot. Then the dealer gets a chance to bet, again either \$0 or \$1. Finally, if the gambler bet \$0 and the dealer bet \$1, the gambler gets a second chance to bring her bet up to \$1. If either player bets \$0 when the other has already bet \$1, that player folds and loses her ante. If neither player folds, the higher card wins the pot, resulting in a net gain of either \$1 or \$2 (equal to the other player's ante plus the bet of \$0 or \$1). In contrast to the usual practice in poker we assume that the payoff vector $c_t$ is observable after each hand; the partially-observable extension is beyond the scope of this paper.

One-card poker is a simple game; nonetheless it has many of the elements of more complicated games, including incomplete information, chance events, and multiple stages. And, optimal play requires behaviors like randomization and bluffing. The biggest strategic difference between one-card poker and larger variants such as draw, stud, or hold-em is the idea of hand potential: while

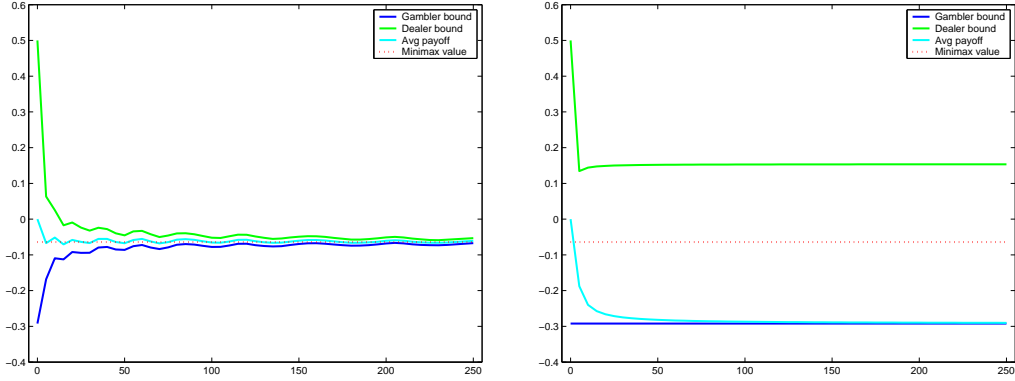

Figure 3: Performance in self-play (left) and against a fixed opponent (right).

45679 and 24679 are almost equally strong hands in a showdown (they are both 9-high), holding 45679 early in the game is much more valuable because replacing the 9 with either a 3 or an 8 turns it into a straight.

Fig. 3 shows the results of two typical runs: in both panels the dealer is using our no-regret algorithm. In the left panel the gambler is also using our no-regret algorithm, while in the right panel the gambler is playing a fixed policy. The $x$-axis shows number of hands played; the $y$-axis shows the average payoff per hand from the dealer to the gambler. The value of the game, $-\$0.064$, is indicated with a dotted line. The middle solid curve shows the actual performance of the dealer (who is trying to minimize the payoff).

The upper curve measures the progress of the dealer's learning: after every fifth hand we extracted a strategy $y_t^{\mathrm{avg}}$ by taking the average of our algorithm's predictions so far. We then plotted the worst-case value of $y_t^{\mathrm{avg}}$. That is, we plotted the payoff for playing $y_t^{\mathrm{avg}}$ against an opponent which knows $y_t^{\mathrm{avg}}$ and is optimized to maximize the dealer's losses. Similarly, the lower curve measures the progress of the gambler's learning.

In the right panel, the dealer quickly learns to win against the non-adaptive gambler. The dealer never plays a minimax strategy, as shown by the fact that the upper curve does not approach the value of the game. Instead, she plays to take advantage of the gambler's weaknesses. In the left panel, the gambler adapts and forces the dealer to play more conservatively; in this case, the limiting strategies for both players are minimax.

The curves in the left panel of Fig. 3 show an interesting effect: the small, damping oscillations result from the dealer and the gambler "chasing" each other around a minimax strategy. One player will learn to exploit a weakness in the other, but in doing so will open up a weakness in her own play; then the second player will adapt to try to take advantage of the first, and the cycle will repeat. Each weakness will be smaller than the last, so the sequence of strategies will converge to a minimax equilibrium. This cycling behavior is a common phenomenon when two learning players play against each other. Many learning algorithms will cycle so strongly that they fail to achieve the value of the game, but our regret bounds eliminate this possibility.

## 9  Discussion

We have presented the Lagrangian Hedging algorithms, a family of no-regret algorithms for OCP based on general potential functions. We have proved regret bounds for LH algorithms and demonstrated experimentally that these bounds lead to good predictive performance in practice. The regret bounds for LH algorithms have low-order dependences on $d$, the number of dimensions in the hypothesis set $\mathcal{Y}$. This low-order dependence means that the LH algorithms can learn well in prediction problems with complicated hypothesis sets; these problems would otherwise require an impractical amount of training data and computation time.

Our work builds on previous work in online learning and online convex programming. Our contributions include a new, deterministic algorithm; a simple, general proof; the ability to build algorithms from a more general class of potential functions; and a new way of building good potential functions from simpler hedging functions, which allows us to construct potential functions for arbitrary convex hypothesis sets. Future work includes a no-internal-regret version of the LH algorithm, as well as a bandit-style version. The former will guarantee convergence to a correlated equilibrium in nonzero-sum games, while the latter will allow us to work from incomplete observations of the cost vector (*e.g.*, as might happen in an extensive-form game such as poker).

**Acknowledgments** Thanks to Amy Greenwald, Martin Zinkevich, and Sebastian Thrun, as well as Yoav Shoham and his research group. This work was supported by NSF grant EF-0331657 and DARPA contracts F30602-01-C-0219, NBCH-1020014, and HR0011-06-0023. The opinions and conclusions are the author's and do not reflect those of the US government or its agencies.

## Footnotes

[1]Many problems use loss functions of the form $\ell_t(y_t) = \ell(y_t, y_t^{\text{true}})$, where $\ell$ is a fixed function such as squared error and $y_t^{\text{true}}$ is a target output. The more general notation allows for problems where there may be more than one correct prediction.

[2]Eq. (7) is similar to the definition of the convex dual $W^*$, but the supremum is over $\bar{y} \in \bar{\mathcal{Y}}$ instead of over all $\bar{y}$. As a result, $F$ and $W^*$ can be very different functions. As discussed in [6], $F$ can be expressed as the dual of a function related to $W$.

# References

[1] Geoffrey J. Gordon. *Approximate Solutions to Markov Decision Processes*. PhD thesis, Carnegie Mellon University, 1999.

[2] James F. Hannan. Approximation to Bayes risk in repeated play. In M. Dresher, A. Tucker, and P. Wolfe, editors, *Contributions to the Theory of Games*, volume 3, pages 97–139. Princeton University Press, 1957.

[3] Martin Zinkevich. Online convex programming and generalized infinitesimal gradient ascent. In *Proceedings of the Twentieth International Conference on Machine Learning*. AAAI Press, 2003.

[4] Adam Kalai and Santosh Vempala. Geometric algorithms for online optimization. Technical Report MIT-LCS-TR-861, Massachusetts Institute of Technology, 2002.

[5] Yoav Freund and Robert E. Schapire. A decision-theoretic generalization of on-line learning and an application to boosting. In *EuroCOLT 95*, pages 23–37. Springer-Verlag, 1995.

[6] Geoffrey J. Gordon. No-regret algorithms for structured prediction problems. Technical Report CMU-CALD-05-112, Carnegie Mellon University, 2005.

[7] Nicolò Cesa-Bianchi and Gábor Lugosi. Potential-based algorithms in on-line prediction and game theory. *Machine Learning*, 51:239–261, 2003.

[8] David Blackwell. An analogue of the minimax theorem for vector payoffs. *Pacific Journal of Mathematics*, 6(1):1–8, 1956.

[9] Shai Shalev-Shwartz and Yoram Singer. Convex repeated games and Fenchel duality. In B. Schölkopf, J.C. Platt, and T. Hofmann, editors, *Advances in Neural Information Processing Systems*, volume 19, Cambridge, MA, 2007. MIT Press.

[10] David P. Helmbold and Robert E. Schapire. Predicting nearly as well as the best pruning of a decision tree. In *Proceedings of COLT*, pages 61–68, 1995.

[11] Eiji Takimoto and Manfred Warmuth. Path kernels and multiplicative updates. In *COLT*, 2002.

[12] R. Tyrell Rockafellar. *Convex Analysis*. Princeton University Press, New Jersey, 1970.

[13] Nick Littlestone and Manfred Warmuth. The weighted majority algorithm. Technical Report UCSC-CRL-91-28, University of California Santa Cruz, 1992.

[14] Sergiu Hart and Andreu Mas-Colell. A simple adaptive procedure leading to correlated equilibrium. *Econometrica*, 68(5):1127–1150, 2000.

[15] H. Brendan McMahan, Geoffrey J. Gordon, and Avrim Blum. Planning in the presence of cost functions controlled by an adversary. In *Proceedings of the Twentieth International Conference on Machine Learning*, 2003.

[16] David P. Helmbold and Robert E. Schapire. Predicting nearly as well as the best pruning of a decision tree. In *COLT*, 1995.

[17] D. Koller, N. Meggido, and B. von Stengel. Efficient computation of equilibria for extensive two-person games. *Games and Economic Behaviour*, 14(2), 1996.
